# Dissect Black Box: Interpreting for Rule-Based Explanations in Unsupervised Anomaly Detection

**Yu Zhang**$^{♭♯§*}$, **Ruoyu Li**$^{♯¶*}$, **Nengwu Wu, Qing Li**$^{♯†}$, **Xinhan Lin**$^{♭}$, **Yang Hu**$^{§}$, **Tao Li**$^{△}$, **Yong Jiang**$^{♭}$

$^{♭}$Shanghai Artificial Intelligence Laboratory, China; $^{♯}$Peng Cheng Laboratory, China
$^{¶}$College of Computer Science and Software Engineering, Shenzhen University, China
$^{§}$Tsinghua University, China; $^{♮}$Tsinghua Shenzhen International Graduate School, China;
$^{△}$Hunan University of Science and Technology, China
yu-zhang23@mails.tsinghua.edu.cn; liruoyu0401@outlook.com; wnw@mail.hnust.edu.cn
liq@pcl.ac.cn; hu_yang@tsinghua.edu.cn; tlee@hnust.edu.cn; linxinhan@pjlab.org.cn;
jiangy@sz.tsinghua.edu.cn;

## Abstract

In high-stakes sectors such as network security, IoT security, accurately distinguishing between normal and anomalous data is critical due to the significant implications for operational success and safety in decision-making. The complexity is exacerbated by the presence of unlabeled data and the opaque nature of black-box anomaly detection models, which obscure the rationale behind their predictions. In this paper, we present a novel method to interpret the decision-making processes of these models, which are essential for detecting malicious activities without labeled attack data. We put forward the Segmentation Clustering Decision Tree (SCD-Tree), designed to dissect and understand the structure of normal data distributions. The SCD-Tree integrates predictions from the anomaly detection model into its splitting criteria, enhancing the clustering process with the model's insights into anomalies. To further refine these segments, the Gaussian Boundary Delineation (GBD) algorithm is employed to define boundaries within each segmented distribution, effectively delineating normal from anomalous data points. At this point, this approach addresses the curse of dimensionality by segmenting high-dimensional data and ensures resilience to data variability and perturbations through flexible boundary fitting. We transform the intricate operations of anomaly detection into an interpretable rule's format, constructing a comprehensive set of rules for understanding. Our method's evaluation on diverse datasets and models demonstrates superior explanation accuracy, fidelity, and robustness over existing method, proving its efficacy in environments where interpretability is paramount.

## 1 Introduction

Anomaly detection, particularly in its unsupervised form, holds significant promise by leveraging advanced machine learning to identify outliers or unusual patterns without labeled datasets [1] in the ML and DL domains. However, this promise is often hindered by the "black-box" nature of many models, which, while adept at detecting anomalies, provide little insight into the "why" behind their judgments [2]. This opacity poses a substantial barrier to trust between AI systems and human experts, fostering uncertainty and potentially leading to delays or errors, thereby undermining the reliability and efficiency these systems are designed to enhance [3].

---

$^{*}$These authors contributed equally to this work.
$^{†}$Corresponding author: Qing Li.

To address the trust requirements in high-stakes sectors like cybersecurity [4, 5], rule-based surrogate models [6, 7] have been proposed to provide interpretability for data by approximating their decision-making processes, such as decision trees [8, 9] and linear models [10], which are effective in simpler scenarios. However, these surrogate models often struggle to imitate the complexity of black-box models, especially with high-dimensional data, leading to oversimplified explanations that may not faithfully represent the original model's behavior.

**Curse of Dimensionality.** The challenge of scalability in the context of high-dimensional data is a critical concern in anomaly detection [11, 12], where the "curse of dimensionality" can significantly obscure meaningful patterns and exacerbate computational demands. In addition, data that appears to belong to a single category might actually embody several underlying distributions, which may cause model misunderstandings. Higher dimensional spaces are inherently sparse, and the volume of the space grows exponentially with dimension, leading to difficulties in clustering, and discretizing the space and processing it in a distribution space is by far the best solution.

**Inresilliance to data variability.** Resilience to data variability and perturbations is a crucial aspect of robust anomaly detection. While some methods [13, 14] often lack the flexibility to adapt data patterns to the flexible distribution of data, and they determine boundaries that are too "hard" to adequately fit the expanded boundaries to predictions of unknown data, which is essential for maintaining the reliability and robustness of anomaly detection systems in dynamic environments.

Our work revolves around extracting rules that can globally explain the model's decisions which is done through two main contributions: distribution decomposition rules via the Segmentation Clustering Decision Tree (SCD-Tree) and the Gaussian Boundary Delineation (GBD) algorithm. The process begins with the creation of SCD-Tree, an unsupervised tree model that uses the block-box model's predictions as criteria for splitting the data into clusters, each representing a segment of the data distribution. This approach helps manage the curse of dimensionality by breaking down the high-dimensional space into more manageable segments. Within these segmented areas, GBD offers a significant advantage by focusing on the most relevant dimensions to model the decision boundary. This allows the GBD algorithm to simulate boundaries effectively, ensuring resilience against data variability and perturbations, and achieving robust interpretation.

In the experimental phase, we meticulously tested the capabilities of our model, focusing on its ability to autonomously extract rules from black-box anomaly detection models and evaluate its accuracy in detecting anomalies. This work focuses on structured, tabular data, which is prevalent in many high-stakes domains such as network and IoT security. To rigorously assess the effectiveness and reliability of our approach, we conducted extensive comparisons against five established baseline anomaly detection models across four diverse datasets from network security, IoT security, and application security, focusing on structured, tabular data prevalent in these high-stakes domains.. The results demonstrate that our method not only excels in extracting interpretable and precise rules from black-box models but also outperforms in terms of fidelity, robustness, and detection rates (true positive and true negative rates). This performance underscores the utility of our model in enhancing trust in automated systems and confirms its suitability for high-stakes environments where high accuracy in anomaly detection is paramount for effective deployment.

## 2   Related Work

**Unsupervised anomaly detection** does not require labeled data, making it highly adaptable and efficient for detecting novel or unknown types of anomalies in environments, which includes methods such as clustering-based approaches [15] like K-means [16, 17] and DBSCAN [18, 19], which assume normal data points cluster together while anomalies do not. Isolation Forest [20, 21], stands out by isolating anomalies instead of profiling normal data, providing scalability and efficiency. More recently, neural network-based methods, especially autoencoders [22, 23], have gained prominence due to their ability to learn complex, non-linear data representations. However, they are often criticized for their "black-box" nature, which impedes the interpretability of their decisions [24].

**Model Interpretability.** Enhancing the transparency of anomaly detection models involves techniques ranging from model-agnostic methods like LIME [25] and SHAP [26] to model-specific adjustments that incorporate interpretability directly into the model architecture [27, 28, 29], and decision trees or simpler linear models, are trained to approximate the predictions of the original complex models. These explanatory models using local post-hoc techniques [30, 31] attempt to gen-

eralize across different models by approximating the decision function of the black-box model with inherently interpretable structures. However, the ability of these explanatory models to generalize effectively across different types of black-box models can vary considerably. The complexity of the black-box model's structure and the data it handles can limit the fidelity of such explanations, leading to oversimplifications that fail to capture crucial nuances in decision-making.

In addition, these models often assume linear relationships between features [32, 33], which is rarely the case in complex datasets with non-linear interactions, leading them to potentially oversimplify data complexities and result in incomplete or inaccurate interpretations [34]. They are fundamentally simplifications of the original models, and as such, they may not capture all the nuances of complex decision boundaries effectively, which may lead to a loss of fidelity, where the surrogate model's predictions do not consistently align with those of the black-box model [35].

**Curse of dimensionality.** The "curse of dimensionality" in high-dimensional data significantly complicates anomaly detection by obscuring meaningful patterns and relationships. Recent research has proposed various strategies to mitigate its effects, including feature selection [36, 37, 38] and extraction [39] techniques that identify the most informative features of the dataset. In addition, various learning techniques such as t-SNE and UMAP have been used to visualize and understand complex data structures. While these methods are invaluable for simplifying high-dimensional data into more manageable forms, they can sometimes strip away nuances of the original data structure. This oversimplification can result in the loss of information that is critical to accurately identifying anomalies, as certain anomalies may only be detectable in the original high-dimensional space.

**Dynamic Updateability.** Additionally, these surrogate models [13, 40, 14] often assume static data distributions and fail to adapt to the dynamic nature of real-world data, where new anomaly patterns can emerge over time. This assumption limits their effectiveness in continuously evolving environments, where the ability to adapt and update explanations in response to new data is crucial.

In summary, despite black-box models offer significant potential in unsupervised anomaly detection, their application is hindered by a lack of interpretability, reliance on oversimplified surrogate models, and challenges in handling dynamic, high-dimensional data. Addressing these limitations necessitates the development of advanced explanation methodologies that maintain robustness to original models' decisions while providing clear, comprehensive, and adaptable insights into their operational logic.

# 3 Overview

## 3.1 Problem Definition

**Definition 1 (Unsupervised Anomaly Detection Framework):** Let $\mathbf{X} \subset \mathbb{R}^d$ assume unlabeled data sampled from a stationary distribution $\mathcal{D}$, representing a $d$-dimensional feature space. A vector $\mathbf{x} \in \mathbf{X}$ represents a feature vector with components $\mathbf{x} = (x_1, \ldots, x_d)$. The detection model is tasked with estimating a density function $\hat{p}(\mathbf{x})$, which approximates the true underlying distribution $p(\mathbf{x})$. Anomalies are detected based on the density threshold $\varphi$, defined as: $p(\mathbf{x}) > \varphi \Rightarrow$ "$Anomaly''$, which can identify data points that deviate significantly from established patterns of 'normality'.

Threshold $\varphi(>0)$ serves as the critical boundary between normal and abnormal data, deliberately set above zero to account for the inherent false positive rates—often minimized but inevitable in anomaly detection models as discussed in prevailing studies [41, 42, 43]. This setup acknowledges occasional data contamination and errors, ensuring the model's robustness to a small proportion of noisy data without relying on ground truth labels for training.

**Definition 2 (Rule-Based Interpretative Model):** To elucidate the decision-making process of the anomaly detection model, we construct a rule-based interpretative model. This model partitions the data space into a series of interpretable regions using a set of logical rules.

**Rule Construction:** A rule set $\mathcal{R} = R_1, R_2, \ldots, R_k$ is formed by a conjunction of linear inequalities where each rule $R$ captures specific characteristics of the data distribution as understood by surrogate model $f$. Each rule $R$ is a logical conjunction of conditions on the feature dimensions, defined as $R = \bigwedge_{i=1}^{d}(s_i \cdot x_i \odot_i \varphi_i)$, where $\odot_i \in \leq, >$ is relational operator, comparing the scaled feature value $s_i \cdot x_i$ to the threshold $\varphi_i$. The rule is satisfied if all inequalities hold true for every dimension.

**Surrogate Model Definition:** To evaluate a data point as anomalous if it does not satisfy any rule within $\mathcal{R}$, the surrogate model $h_{\mathcal{R}}(\mathbf{x})$ is constructed from the rule set $\mathcal{R}$: $h_{\mathcal{R}}(\mathbf{x}) = \neg \bigwedge_{R \in \mathcal{R}}(x \in R)$. This formulation enables the surrogate model to identify anomalies by their failure to conform to any of the normal profiles described by $\mathcal{R}$.

**Our goal:** To ensure that the rule set $\mathcal{R}$ extracted from the anomaly detection model accurately mirrors the behavior of the underlying model, with an emphasis on the true positive rate and true negative rate. We refine $\mathcal{R}$ to minimize the divergence between the predictions of the anomaly detection model and the classifications made by $h_{\mathcal{R}}(\mathbf{x})$. This is formalized as follows:

$$\arg\min_{\mathcal{R}} \mathcal{L}(\mathcal{R}, \hat{p}, \varphi) = \arg \bigcup_{i=1}^{k} \mathcal{R}_{i|I(\hat{p}(x_i)<\varphi)-I(h_{\mathcal{R}}(x_i)='Anomaly')|} \quad (1)$$

where $I$ denotes the indicator function, and $\mathcal{L}$ measures the mismatch between model predictions and rule-based classifications over the entire data space.

## 3.2 Methodology Overview

In addressing the intrinsic complexities of unsupervised anomaly detection, especially in handling high-dimensional, multimodal data distributions, our methodology embraces a divide-and-conquer strategy, tailored to dissect and understand these distributions through a robust, structured approach. Leveraging the capabilities of the Segmentation Clustering Decision Tree (Section 4) and Gaussian Boundary Delineation (Section 5), our methodology evolves to refine boundaries.

### 3.2.1 Multimodal Data and Initial Segmentation

Recognizing the multimodal nature of typical datasets in applications such as network security and healthcare - where data points are derived from various normal operations that inherently form distinct clusters in the feature space, which SCD-Tree achieves the strategic segmentation:

$$\bigoplus_{k=1}^{K} \arg\min_{R_k} \left[\alpha \cdot \mathcal{L}_{\mathcal{X} \in \mathcal{D}_k}(R_k, f, \varphi) + \beta \cdot \mathcal{L}_{\mathcal{X} \notin \mathcal{D}_k}(R_k, f, \varphi)\right], \text{Where } R_k = \text{AND}(R_k^I, R_k^E) \quad (2)$$

Here, $\mathcal{L}_{\mathcal{X} \in \mathcal{D}k}$ denotes a hypercube enclosing the $i^{th}$ modal cluster's data samples, facilitating initial data categorization and minimizing the loss $\mathcal{L}_{\mathbf{x} \in \mathbf{X}}(\mathcal{R}, f, \varphi)$ for in-distribution points.

### 3.2.2 Boundary Delineation

For each $i$ using GBD based on subspaces to cover maximal normal data while excluding outliers. This step involves dynamically adjusting the decision boundaries within each identified segment using Gaussian Processes to model the decision boundaries accurately. The Gaussian Processes provide a probabilistic framework that not only defines the boundaries but also quantifies the uncertainty in these boundaries, thereby enhancing the model's interpretative power.

Finally, we get a comprehensive rule set $[\mathcal{R} = \bigcup_{i=1}^{m} \text{RefinedRules}(H_i, \mathcal{D}_i)]$ by synthesizing the refined rules from each segment that globally approximates the behavior of the original anomaly detection model across the entire dataset. This synthesized rule set not only ensures high fidelity in anomaly detection but also facilitates a deeper understanding of the model's decision-making process, crucial for trust and transparency in critical applications.

## 3.3 Anomaly detection pipline

As Figure 1 shows, The initial datasets are transformed through normalization and scaling processes to ensure uniformity and mitigate the influence of outlier values.

**Unsupervised Tree Segmentation:** The SCD-Tree utilizes original detection model outputs to guide its data segmentation process. By integrating anomaly scores (e.g., MSE, Probability) directly into the tree's branching logic, the SCD-Tree adapts its segmentation boundaries dynamically, reflecting

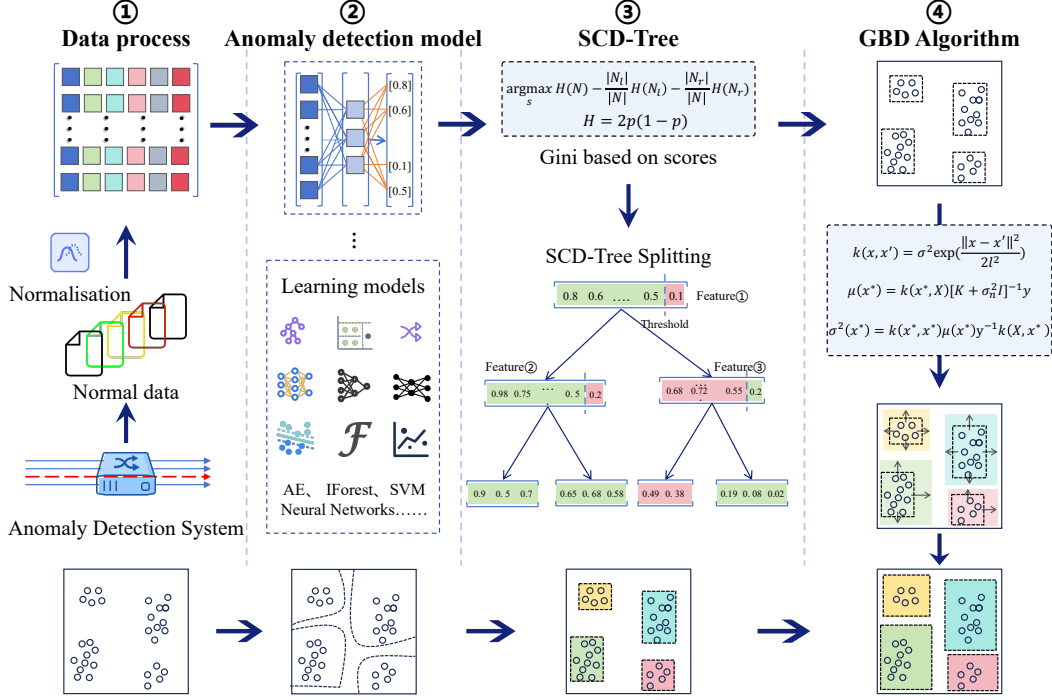

Figure 1: Integrated Anomaly Detection and Interpretation Pipeline Using SCD-Tree and GBD

subtle shifts in data patterns and anomaly distributions. This integration allows the SCD-Tree to partition the dataset into clusters that are homogeneously normal, yet distinct from one another based on the learned anomaly characteristics.

**Gaussian Boundary Delineation:** Following the data segmentation by the SCD-Tree, the Gaussian Boundary Delineation is employed to define and refine the decision boundaries between identified clusters. This approach uses Gaussian Processes to achieve a fine-grained mapping of decision boundaries, accommodating the non-linear and complex nature of the data structures encountered.

**Rule Extraction and Interpretation:** Once the boundaries are established, the GBD algorithm translates these complex mathematical models into a set of clear, interpretable rules. Each rule delineates conditions under which data points are considered normal, in the subsequent judgment, we only need to judge whether the data meets the RULE criteria for normal data based on the characteristics of the data, if it meets it is judged to be normal data, otherwise it is considered to be abnormal data. This offers actionable insights into the underlying decision-making process of the anomaly detection model.

**Proposition.** *Let* $\mathbf{X} \subset \mathbb{R}^d$ *be a dataset sampled from a stationary distribution* $\mathcal{D}$*, and let* $f : \mathbf{X} \to \{0, 1\}$ *be an unsupervised anomaly detection model. There exists a rule-based surrogate model* $h_{\mathcal{R}}(\mathbf{x})$ *derived from* $f$ *such that* $h_{\mathcal{R}}$ *maintains high fidelity to* $f$*, achieving high true negative rate (TNR) and true positive rate (TPR).*

# 4 Segmentation Clustering Decision Tree

In the quest to unravel the complex decision-making mechanisms of unsupervised anomaly detection models, our methodology adopts a sophisticated, dual-faceted approach that systematically dissects and models the data distribution $\mathcal{D}$. Recognizing the inherent multimodality and high dimensionality of typical datasets in domains like network security, IoT security, we commence with the Segmentation Clustering Decision Tree to partition the feature space into distinct subspaces.

The foundational principle of the SCD-Tree is based on the hypothesis that data points with similar anomaly scores likely share the same operational conditions or states. This hypothesis is formalized

in the tree's splitting criteria, which aim to minimize within-cluster variance of anomaly scores while maximizing between-cluster differences, thus ensuring each cluster is as homogeneous as possible.

Based on this, SCD-Tree extends the CART decision tree [44] by integrating black-model-based predictions into its decision-making process. Unlike traditional decision trees which rely on labeled data for node splitting, the SCD-Tree harnesses the original model outputs and threshold to segment data into coherent subsets that reflect underlying distribution patterns.

**Node Splitting.** For each node within SCD-Tree, the splitting process is governed by the distribution of anomaly detection model outputs, $f(\boldsymbol{x})$ across data samples. A node $\boldsymbol{N}$ splits based on the criterion that maximizes the difference in model output distributions between subsets, formalized as:

$$s = \arg\max_s H(\boldsymbol{N}) - \frac{|\boldsymbol{N}_l|}{|\boldsymbol{N}|} H(\boldsymbol{N}_l) - \frac{|\boldsymbol{N}_r|}{|\boldsymbol{N}|} H(\boldsymbol{N}_r) \qquad (3)$$

where $s = (i, \phi_i)$ is the splitting condition, $\phi_i$ being the threshold for feature $i$, and $H$ represents a homogeneity score based on the output of the anomaly detection model: $H = 2p(1-p)$, with $p_k$ as the proportion of samples in node $\boldsymbol{N}$ that fall into the $k$-th output category of the anomaly model.

**Termination Criteria.** Splitting continues recursively until any of the following conditions is met: i) the node $\boldsymbol{N}$ contains only one sample, ii) the variance in anomaly detection scores within a node is less than $\epsilon$, or iii) the tree reaches a predetermined maximum depth $\tau$.

**Distribution Decomposition Rule Extraction.** Upon the completion of the SCD-Tree training, the resulting structure facilitates a granular understanding of the data through its segmented distributions, each represented by a leaf node. The transformation of these segments into a set of operational rules provides a systematic method for categorizing new data points based on the learned segments.

For each leaf node, the sequential decisions from the root compile into a cohesive rule set, outlining the conditions for data points within the corresponding segment:

$$R_k = \bigcap_{h=1}^{\tau} \{x_{f_h} \circ_h \varphi_h\} \qquad (4)$$

where $R_k$ denotes the rule set corresponding to the $k$-th segment. Here, $\tau$ is the depth of the leaf node, indicating the number of conditions (or splits) from the root to the leaf. The variable $f_h$ identifies the feature upon which the split is made at the $h$-th level of the tree, $\varphi_h$ is the threshold value for that split, and $\circ_h$ represents the relational operator (either "$\leq$" for a left split or '$>$' for a right split).

The culmination of the SCD-Tree training results in a multifaceted representation of the data's structure, captured by each leaf node's unique profile, and traversing from the root to a specific leaf encapsulates a precise rule pathway that partitions the feature space into distinct subspaces $R_k$.

# 5 Gaussian Process for Boundary Delineation

Following the creation of these distinct subspaces $R_k$ through the SCD-Tree, we employ *Gaussian Process for Boundary Delineation* within each subspace to meticulously define and refine the decision boundaries, capturing the subtle variations and dynamics of the data to enhance the precision of anomaly detection.

**Boundary Estimation Process.** Within the Gaussian Process (GP) framework, consider a set of data points $X_{\text{in}}$ classified as normal by the SCD-Tree, along with their corresponding scores $s$. The binary classification outputs $y$ are determined by comparing these scores to a predefined threshold $\varphi$. Specifically, $y$ is defined as $y_i = 1$ if $s_i \leq \varphi$ (indicating normality) and $y_i = 0$ if $s_i > \varphi$ (indicating anomaly). This classification serves as input to the GP, which provides a predictive distribution for any new point $\mathbf{x}^*$ based on these inputs:

$$p(f(\mathbf{x}^*)|\mathbf{X}_{\text{in}}, \mathbf{y}, \mathbf{x}^*) = \mathcal{N}(\mu(\mathbf{x}^*), \sigma^2(\mathbf{x}^*)) \qquad (5)$$

where $\mu(\mathbf{x})$ and $\sigma^2(\mathbf{x}^*)$ represent the mean and variance of the predictive distribution, respectively, and $f(\mathbf{x}) \sim \mathcal{GP}(m(\mathbf{x}), k(\mathbf{x}, \mathbf{x}'))$. This probabilistic framework facilitates dynamic assessment of whether new data points conform to the patterns of normality or are indicative of anomalies.

The mean $\mu(\mathbf{x}^*)$ and variance $\sigma^2(\mathbf{x}^*)$ of this distribution are given by:

$$\mu(\mathbf{x}^*) = \mathbf{k}^T[\mathbf{K} + \sigma_n^2\mathbf{I}]^{-1}\mathbf{y}$$
$$\sigma^2(\mathbf{x}^*) = k(\mathbf{x}^*, \mathbf{x}^*) - \mathbf{k}^T[\mathbf{K} + \sigma_n^2\mathbf{I}]^{-1}\mathbf{k} \tag{6}$$

where $\mathbf{k}$ is the vector of covariances between $\mathbf{x}^*$ and each point in $\mathbf{X}_{\text{in}}$, $\mathbf{K}$ is the covariance matrix for $\mathbf{X}_{\text{in}}$, which is computed as $k(x, x^*) = \sigma^2 exp(-\frac{\|x-x^*\|^2}{2l^2})$, and $\sigma_n^2$ is the noise variance, capturing inherent data uncertainties.

**Defining the Decision Boundaries.** The essential step in boundary delineation is to identify the set $\Omega$ where $\mu(\mathbf{x})$ surpasses a predefined threshold $\theta$ (indicating high confidence in normality) and $\sigma(\mathbf{x})$ is minimal (indicating low uncertainty). These criteria ensure robust and precise boundary definition:

$$\Omega = \{\mathbf{x} \in \mathbb{R}^d : \mu(\mathbf{x}) > \theta \wedge \sigma(\mathbf{x}) < \epsilon\} \tag{7}$$

Once $\mu(\mathbf{x})$ and $\sigma(\mathbf{x})$ are obtained, we use the data points $X_k$ within the subspace $D_k$ and the extended set of data points $X_k'$ sampled around it to capture boundary variations. The final boundary can be obtained from the following equation:

$$\mathcal{R}_k = [min(\mu(X_k) - \sigma^2(X_k)), max(\mu(X_k') + \sigma^2(X_k'))] \tag{8}$$

**Rule Acquisition and Interpretation.** For each dimension $j$ of the feature space, a rule is derived based on these boundaries: $R_j : \theta_{\text{low},j} \leq x_j \leq \theta_{\text{high},j}$, where $\theta_{\text{low},j}$ and $\theta_{\text{high},j}$ are the lower and upper bounds of the decision boundary for feature $j$, computed from the GP outputs. Finally, the rules $R_j$ are then synthesized into a comprehensive rule set $\mathcal{R}$, defined as: $\mathcal{R} = \bigcap_{j=1}^{d} R_j$, which globally approximates the behavior of the original anomaly detection model across the entire dataset. This rule set classifies a data point $\mathbf{x}$ as normal if it satisfies all individual feature rules, thus aligning with the patterns of normality as modeled by the Gaussian Processes within their respective subspaces.

**Rule Refinement.** To ensure that the rules are robust and reflect the subtle nuances of the data, they are continuously refined based on feedback from ongoing anomaly detection operations. This dynamic refinement process utilizes new data to adjust the thresholds $\theta_{\text{low},j}$ and $\theta_{\text{high},j}$, thereby adapting the rule set $\mathcal{R}$ to evolving data conditions and anomaly patterns.

# 6 Evaluation

## 6.1 Experimental Setup

**Datasets.** We employ four distinct datasets to evaluate our method across various security-related domains. These datasets include Malicious and Benign Webpages [45] for web security, KDDCup [46] for classic network intrusion scenarios [47], CIC-IDS [48] for modern network attacks, and TON-IoT [49], which integrates IoT and traditional network data.

These tabular format datasets are systematically partitioned into training, validation, and testing segments following an 8:1:1 ratio split. For the calibration of our anomaly detection models' hyperparameters, only normal instances from these datasets are utilized. The performance metrics, specifically the accuracy for each model, are comprehensively documented in Table 1.

**Black Models.** For model comparison, we utilize four established unsupervised models: Autoencoders (AE [41]) and Variational Autoencoders (VAE [50]) detect anomalies through reconstruction errors; One-Class SVM (OCSVM [51]) isolates normal data in feature space; and Isolation Forest (IForest [52]) efficiently identifies outliers. These models facilitate a robust assessment of our method's effectiveness across diverse data scenarios. In addition to these classical methods, more advanced models such as Variational Recurrent Autoencoder (VRAE [53]), Deep Autoencoding Gaussian Mixture Model (DAGMM [54]) are evaluated. These advanced models facilitate a robust

Table 1: The accuracy for four anomaly detection models applied to datasets in security.

| No. | Dataset | #Domain | #Features | #Normal | #Attack | AE | VAE | OCSVM | iForest |
|-----|---------|---------|-----------|---------|---------|-----|-----|-------|---------|
| 1 | Webpages | Web Security | 10 | 98% | 2% | 0.9999 | 0.9975 | 0.998 | 0.9925 |
| 2 | TON-IoT | IoT Security | 30 | 25.71% | 74.29% | 0.9839 | 0.9886 | 0.9957 | 0.9943 |
| 3 | Kddcup99 | Cybersecurity | 41 | 19.86% | 80.14% | 0.9895 | 0.944 | 0.9415 | 0.9905 |
| 4 | CIC-IDS | Cybersecurity | 80 | 70.45% | 29.55% | 0.9438 | 0.9251 | 0.9706 | 0.9972 |

assessment of our method's effectiveness across diverse data scenarios, demonstrating its applicability not only to traditional but also state-of-the-art techniques in anomaly detection.

**Baseline Models.** We assess the effectiveness of our approach against several established explanation methods serving as baselines. These include direct rule extraction from unsupervised anomaly detection models (UAD) [55], knowledge distillation techniques (KD) [56] that simplify complex models, and global explanation methods such as greedy decision trees (EGDT) [8], which approximate black-box model decision processes. We also compare with Local Interpretable Model-agnostic Explanations (LIME) [25], providing local insights aggregated into global interpretations, and Trustee [9], which synthesizes local explanations into a cohesive global understanding.

**Metrics.** Our evaluation employs a suite of metrics inspired by [57]: Fidelity, Robustness, both True Positive Rate and True Negative Rate. Their meanings and specific calculations are written in appendix B.4.1. These metrics provide a comprehensive framework for comparing the explanatory power and reliability of different approaches, ensuring that our findings are robust and applicable to real-world anomaly detection scenarios.

## 6.2 Model Interpreting Rationale

To demonstrate the interpretability and effectiveness of our proposed method, we examine rules extracted from a well-trained VAE model applied to different datasets, focusing on Denial of Service (DoS) attacks. As Table 2 indicates, the rule "$ps\_fwd\_var > 101.68$" with an attack value of 57.33 indicates that normal traffic has higher variance in forward packet sizes compared to DoS traffic, which uses consistent sizes to overwhelm targets. Similarly, "$0.949 < iat\_max \leq 7.278$" with an attack value of 0.00063 shows very short intervals between packets, characteristic of DoS attacks. These rules encapsulate DoS behaviors, providing a clear rationale for the model's decisions. Such interpretable rules enhance model transparency, enabling security professionals to understand and respond to cyber threats effectively. This interpretability is crucial in high-stakes environments, ensuring trust and facilitating timely, accurate threat detection.

Table 2: Examples of Explanation for Different Types of Cyber Attacks

| Attack | Rules of Normality | Attack Value | Feature Meaning | Human Understanding |
|---|---|---|---|---|
| **DoS** | $ps\_fwd\_var > 101.68$<br>$0.949 < iat\_max \leq 7.278$<br>$fwd\_count > 124.371$ | 57.33<br>0.00063<br>0.00126 | Variance in forward packet size<br>Maximum inter-arrival time<br>Forward packet count | DoS attacks use consistent small packets at high rates. They also show minimal intervals between packets and high packet counts to overwhelm the network. |
| **MITM** | $count > 92.529$<br>$ps\_var > 67248.034$ | 12<br>82.0 | IP packet count per connection<br>Variance of IP packet sizes | MITM attacks often increase packet count and show high variance in packet sizes due to manipulation. |
| **Ransomware** | $iat\_bwd\_max > 1.1139$<br>$iat\_bwd\_min \leq 0.0262$ | 0.1<br>0.37 | Maximum inter-arrival time<br>Minimum inter-arrival time | Ransomware attacks exhibit irregular traffic patterns and rapid bursts of communication. |
| **Phishing Web** | $url\_cluster \leq 1.1038$<br>$https \leq 0.797$ | 2.0<br>1.0 | URL grouping<br>Uses HTTPS | Phishing websites often group similar URLs and use HTTPS to appear secure and deceive users. |

## 6.3 Precision in Interpretative Output

In our comprehensive assessment, illustrated in Table **??**, we employed four established anomaly detection models—IForest, OCSVM, AE, and VAE—as pre-configured "black models" for our SCD-Tree, and rigorously tested them across four distinct datasets from different security domains. The results demonstrate that our model exhibits robust performance and high effectiveness in identifying anomalies across all domains, with nearly all robustness (RB) and fidelity (FD) metrics surpassing 90%. Characterized by strong discriminative power and fidelity, our findings underscore our model's adaptability and confirm its suitability for diverse anomaly detection tasks in various contexts.

In Tables 3, we evaluate our model with five baselines, it consistently outperforms established baseline methods across a variety of metrics: it achieves True Positive Rates (TPR) ranging from 91.5% to 100%, and True Negative Rates (TNR) as high as 99.62%, showcasing its exceptional capability in accurately identifying both normal and anomalous instances. Moreover, the fidelity (FD) and robustness (RB) metrics impressively exceed 90% in most scenarios, with perfect scores (100%) observed in settings involving the KddCup99 and Web datasets, emphasizing the model's reliable interpretative output under varying operational conditions, proving its resilience to data variability and perturbations. This comprehensive performance underscores the SCD-Tree's efficacy and adaptability,

making it a prime candidate for deployment in high-stakes environments where accuracy, consistency, and interpretability are crucial. More results for metrics are in the appendix.

Table 3: Performance of rule extraction.

(1) Baselines use different Black Model on CIC-IDS Dataset

| Method | AE | | | | VAE | | | | OCSVM | | | | iForest | | | |
|---|---|---|---|---|---|---|---|---|---|---|---|---|---|---|---|---|
| | TPR | TNR | FD | RB | TPR | TNR | FD | RB | TPR | TNR | FD | RB | TPR | TNR | FD | RB |
| UAD | 0.0008 | 0.9786 | 0.1322 | 0.4997 | 0.0213 | 0.9988 | 0.1444 | 0.4834 | 0.9972 | 0.9884 | 0.997 | 0.9997 | 0.0008 | **1.00** | 0.1264 | 0.7009 |
| EGDT | 0.9715 | 0.9715 | 0.4803 | 0.9995 | 0.0799 | 0.995 | 0.0799 | 1.00 | 1.00 | 0.00 | 1.00 | 1.00 | 0.8982 | 0.9392 | 0.8982 | 0.9818 |
| Trustee | 0.3853 | 0.998 | 0.4873 | 0.6408 | 0.0621 | 0.9972 | 0.0621 | 0.9929 | 0.9993 | 1.00 | 0.5389 | 0.6109 | 0.9805 | 0.4485 | 0.4535 | 0.581 |
| LIME | 0.9065 | 0.9786 | 0.9203 | 1.00 | 0.9063 | 0.9786 | 0.941 | 0.9996 | 0.0021 | 0.9005 | 0.013 | 1.00 | 0.8243 | 0.9913 | 0.891 | 1.00 |
| KD | 0.479 | 1.00 | 0.5778 | 0.9988 | 0.1024 | 0.9995 | 0.2001 | 0.9827 | 0.3096 | 1.00 | 0.3621 | 0.9998 | 0.0008 | 1.00 | 0.1264 | 0.7009 |
| Ours | 0.936 | 0.9937 | 0.9431 | 0.9879 | 0.915 | 0.9899 | 0.9453 | 0.8979 | 0.9667 | **0.9962** | 0.9704 | 0.9329 | **0.9973** | 0.9917 | **0.9966** | **0.9956** |

(2) Baselines Use VAE as a Black Model Across Multiple Security Datasets

| Method | CIC-IDS | | | | KddCup99 | | | | TON-IoT | | | | Webpages | | | |
|---|---|---|---|---|---|---|---|---|---|---|---|---|---|---|---|---|
| | TPR | TNR | FD | RB | TPR | TNR | FD | RB | TPR | TNR | FD | RB | TPR | TNR | FD | RB |
| UAD | 0.0213 | 0.9988 | 0.1444 | 0.4834 | 0.0119 | 0.0025 | 0.0119 | 1.00 | 0.0004 | 0.9998 | 0.0485 | 0.4997 | 0.0769 | 0.00 | 0.0769 | 1.00 |
| EGDT | 0.0799 | 0.995 | 0.0799 | 1.00 | 0.9874 | 0.988 | 0.9874 | 1.00 | 0.8048 | 0.9767 | 0.8145 | 1.00 | 0.8378 | 1.00 | 0.8378 | 1.00 |
| Trustee | 0.0621 | 0.9972 | 0.0621 | 0.9929 | 0.2586 | 1.00 | 0.2586 | 0.79 | 0.9998 | 1.00 | 0.7934 | 0.8435 | 0.00 | 0.9995 | 0.00 | 0.9995 |
| LIME | 0.9063 | 0.9786 | 0.941 | 0.9996 | 0.9292 | 0.9808 | 0.9292 | 1.00 | 0.3994 | 0.9643 | 0.3994 | 1.00 | 0.2051 | 0.7909 | 0.2051 | 1.00 |
| KD | 0.1024 | 0.9995 | 0.2001 | 0.9827 | 0.0006 | 0.9973 | 0.0006 | 1.00 | 0.3627 | 0.9997 | 0.3627 | **1.00** | 0.0167 | 0.92 | 0.08 | 1.00 |
| Ours | 0.915 | 0.9899 | 0.9453 | 0.8979 | 0.9971 | 0.9604 | 0.9913 | 0.9983 | 1.00 | 0.9847 | 0.9936 | 0.9197 | 0.898 | **1.00** | 0.998 | **1.00** |

## 6.4 Ablation Experiment

The ablation experiments 4 were designed to systematically evaluate the impact of the Gaussian Boundary Delineation (GBD) on the performance of unsupervised anomaly detection models. The experiment compared models both with and without the incorporation of GBD, when combined with GBD, exhibited a notable increase in true positive rate, rising from 0.9422 to 0.9979, marking an improvement of 0.0557. This significant enhancement indicates that GBD effectively refines decision boundaries, leading to more accurate anomaly detection. Similarly, the fidelity metric improved from 0.928 to 0.9978, underscoring the GBD's role in aligning the model's decision-making process with ground truth data, thereby enhancing the reliability of the model's interpretations.

Table 4: The impact of GBD component on performance.

| Method | TP | FP | FD |
|---|---|---|---|
| VAE | 0.9422 | 0.0035 | 0.928 |
| VAE+GBD | 0.9979(↑ 0.0557) | 0.01 | 0.9978(↑ 0.0698) |
| AE | 0.8122 | 0.0035 | 0.8428 |
| AE+GBD | 0.9356(↑ 0.1234) | 0.0063 | 0.9431(↑ 0.1003) |
| Kmeans | 0.9872 | 0.3456 | – |

## 6.5 Complexity Analysis

We systematically examined the impact of varying the number of features on training time. Our experimental results show a clear increase in training time as the number of features rises. This can be attributed to the growing complexity of the model, as each additional feature expands the data space, necessitating the model to learn a higher-dimensional representation. Consequently, this dimensionality increase leads to more parameters that require optimization, thereby extending the training duration. These findings are consistent with existing literature on model complexity and feature dimensionality. Specific experimental results and explanations are written in appendix A.4.2

However, our model's prediction time remains very fast because it relies on evaluating a set of pre-defined rules derived from the tree's decision paths and boundary delineations, making the process computationally inexpensive and highly efficient for real-time applications.

The complexity of our algorithms hinges on two main components. The SCD-Tree involves sorting operations at each node, which possess a computational complexity of $O(d \cdot N \log N)$ per feature.

Additionally, GBD uses Gaussian Processes, which have a cubic complexity $O(k \cdot n^3)$ due to matrix inversion, with $n$ indicating the data points per segment. Although this complexity might seem high, it's manageable since GP only operates within defined subspaces($\mathcal{D}_k$) post-SCD-Tree segmentation. Post-training, anomaly detection is conducted using a rule set with a fixed complexity of $O(|C| \cdot d)$, where $|C|$ is the rule count and $d$ is the feature size, optimizing both speed and resource in practice.

# 7    Conclusion and Future Work

In addressing the global interpretability challenges of unsupervised anomaly detection, this paper integrates a novel Gaussian Boundary Delineation with Segmentation Clustering Decision Tree to refine and explain the decision-making process of black-box models. This model provides a probabilistic assessment of boundary points, enhancing the interpretative fidelity by quantifying the uncertainty in boundary delineation. The culmination of this process synthesizes a comprehensive rule set, offering a granular yet global perspective on the model's decision-making process, thereby enhancing transparency and trust in automated anomaly detection systems.

Building on the foundational work in model interpretability, future research should pivot towards developing adaptive algorithms capable of dynamically refining and correcting decision processes in real-time, which reduce error rates in black-box models by continuously learning from new data. Additionally, integrating interpretable anomaly detection methods into distributed edge systems, promises to decentralize and accelerate decision-making processes. This could be particularly transformative in sectors where timely, accurate decisions are crucial. Exploring the synergy between lightweight, interpretable models and existing infrastructure could also pave the way for more robust and scalable anomaly detection systems that are both efficient and easier to audit.

## Acknowledgments and Disclosure of Funding

This work is supported by the National Key R&D Program of China under grant No. 2022ZD0162300, the Major Key Project of PCL under grant No. PCL2023A06-4, the National Key Research and Development Program of China under grant No. 2022YFB3105000, and the Shenzhen Key Lab of Software Defined Networking under grant No. ZDSYS201405091729599899.

## Footnotes

[3]The "expanded samples" is "$samples \leftarrow$ extend_scope($X\_input$)" in Algorithm 1

## References

[1] Rui Jiang, Yijia Xue, and Dongmian Zou. Interpretability-aware industrial anomaly detection using autoencoders. *IEEE Access*, 2023.

[2] Ruoyu Li, Qing Li, Yu Zhang, Dan Zhao, Xi Xiao, and Yong Jiang. Genos: General in-network unsupervised intrusion detection by rule extraction. In *IEEE INFOCOM 2024 - IEEE Conference on Computer Communications*, pages 561–570, 2024.

[3] Chun-Hao Chang, Jinsung Yoon, Sercan Ö. Arik, Madeleine Udell, and Tomas Pfister. Data-efficient and interpretable tabular anomaly detection. In *Proceedings of the 29th ACM SIGKDD Conference on Knowledge Discovery and Data Mining, KDD 2023, Long Beach, CA, USA, August 6-10, 2023*, pages 190–201. ACM, 2023.

[4] Jingyu Xiao, Zhiyao Xu, Qingsong Zou, Qing Li, Dan Zhao, Dong Fang, Ruoyu Li, Wenxin Tang, Kang Li, Xudong Zuo, Penghui Hu, Yong Jiang, Zixuan Weng, and Michael R. Lyu. Make your home safe: Time-aware unsupervised user behavior anomaly detection in smart homes via loss-guided mask. In *Proceedings of the 30th ACM SIGKDD Conference on Knowledge Discovery and Data Mining*, page 3551–3562, 2024.

[5] Qingsong Zou, Qing Li, Ruoyu Li, Yucheng Huang, Gareth Tyson, Jingyu Xiao, and Yong Jiang. Iotbeholder: A privacy snooping attack on user habitual behaviors from smart home wi-fi traffic. *Proceedings of the ACM on Interactive, Mobile, Wearable and Ubiquitous Technologies*, 7(1):1–26, 2023.

[6] Xiubin Zhu, Dan Wang, Witold Pedrycz, and Zhiwu Li. Fuzzy rule-based local surrogate models for black-box model explanation. *IEEE Trans. Fuzzy Syst.*, 31(6):2056–2064, 2023.

[7] Martin Flusser and Petr Somol. Efficient anomaly detection through surrogate neural networks. *Neural Comput. Appl.*, 34(23):20491–20505, 2022.

[8] Osbert Bastani, Carolyn Kim, and Hamsa Bastani. Interpreting blackbox models via model extraction. *CoRR*, abs/1705.08504, 2017.

[9] Arthur S. Jacobs, Roman Beltiukov, Walter Willinger, Ronaldo A. Ferreira, Arpit Gupta, and Lisandro Z. Granville. Ai/ml for network security: The emperor has no clothes. In *ACM SIGSAC Conference on Computer and Communications Security (CCS)*, 2022.

[10] Jerome H. Friedman and Bogdan E. Popescu. Predictive learning via rule ensembles. *The Annals of Applied Statistics*, 2(3), 2008.

[11] Hongzuo Xu, Guansong Pang, Yijie Wang, and Yongjun Wang. Deep isolation forest for anomaly detection. *IEEE Trans. Knowl. Data Eng.*, 35(12):12591–12604, 2023.

[12] Xin Zhang, Pingping Wei, and Qingling Wang. A hybrid anomaly detection method for high dimensional data. *PeerJ Comput. Sci.*, 9:e1199, 2023.

[13] Ruoyu Li, Qing Li, Yu Zhang, Dan Zhao, Yong Jiang, and Yong Yang. Interpreting unsupervised anomaly detection in security via rule extraction. In *Advances in Neural Information Processing Systems*, 2023.

[14] Yicheng Zhou, Zhenzhou Lu, Jinghan Hu, and Yingshi Hu. Surrogate modeling of high-dimensional problems via data-driven polynomial chaos expansions and sparse partial least square. *Computer Methods in Applied Mechanics and Engineering*, 364:112906, 2020.

[15] Pavol Mulinka, Pedro Casas, Kensuke Fukuda, and Lukas Kencl. HUMAN - hierarchical clustering for unsupervised anomaly detection & interpretation. In *11th International Conference on Network of the Future*. IEEE, 2020.

[16] Muhammad Imran Faizi and Syed Muhammad Adnan. Improved segmentation model for melanoma lesion detection using normalized cross-correlation-based k-means clustering. *IEEE Access*, 12:20753–20766, 2024.

[17] Meenal Jain, Gagandeep Kaur, and Vikas Saxena. A k-means clustering and SVM based hybrid concept drift detection technique for network anomaly detection. *Expert Syst. Appl.*, 193:116510, 2022.

[18] Martin Ester, Hans-Peter Kriegel, Jörg Sander, and Xiaowei Xu. A density-based algorithm for discovering clusters in large spatial databases with noise. In *Proceedings of the Second International Conference on Knowledge Discovery and Data Mining (KDD-96), Portland, Oregon, USA*, pages 226–231, 1996.

[19] Mohamed Limam El Hairach, Insaf Bellamine, and Amal Tmiri. Anomaly detection in PV modules: A comparative study of dbscan, k-means, isolation forest, and LOF. In *7th IEEE Congress on Information Science and Technology, CiSt 2023, Agadir - Essaouira, Morocco, December 16-22, 2023*, pages 135–139, 2023.

[20] Fei Tony Liu, Kai Ming Ting, and Zhi-Hua Zhou. Isolation forest. In *IEEE International Conference on Data Mining (ICDM)*, 2008.

[21] Hongzuo Xu, Guansong Pang, Yijie Wang, and Yongjun Wang. Deep isolation forest for anomaly detection. *CoRR*, abs/2206.06602, 2022.

[22] Diederik P. Kingma and Max Welling. Auto-encoding variational bayes. In *International Conference on Learning Representations (ICLR)*, 2014.

[23] Ali Nawaz, Shehroz S. Khan, and Amir Ahmad. Ensemble of autoencoders for anomaly detection in biomedical data: A narrative review. *IEEE Access*, 12:17273–17289, 2024.

[24] Selim F. Yilmaz and S. Kozat. Unsupervised anomaly detection via deep metric learning with end-to-end optimization. *ArXiv*, abs/2005.05865, 2020.

[25] Marco Tulio Ribeiro, Sameer Singh, and Carlos Guestrin. "why should i trust you?": Explaining the predictions of any classifier. In *ACM SIGKDD International Conference on Knowledge Discovery and Data Mining (KDD)*, 2016.

[26] Scott M. Lundberg and Su-In Lee. A unified approach to interpreting model predictions. In *Annual Conference on Neural Information Processing Systems (NeurIPS)*, 2017.

[27] Zhifeng Kong and Kamalika Chaudhuri. Understanding instance-based interpretability of variational auto-encoders. In *Annual Conference on Neural Information Processing Systems (NeurIPS)*, 2021.

[28] Jonathan Crabbé and Mihaela van der Schaar. Label-free explainability for unsupervised models. In *International Conference on Machine Learning (ICML)*, 2022.

[29] Oliver Eberle, Jochen Büttner, Florian Kräutli, Klaus-Robert Müller, Matteo Valleriani, and Grégoire Montavon. Building and interpreting deep similarity models. *IEEE Trans. Pattern Anal. Mach. Intell.*, 44(3):1149–1161, 2022.

[30] Andrea Agiollo, Luciano Cavalcante Siebert, Pradeep Kumar Murukannaiah, and Andrea Omicini. The quarrel of local post-hoc explainers for moral values classification in natural language processing. In *Explainable and Transparent AI and Multi-Agent Systems*, volume 14127 of *Lecture Notes in Computer Science*, 2023.

[31] Vlad Miron, Flavius Frasincar, and Maria Mihaela Trusca. Explaining a deep learning model for aspect-based sentiment classification using post-hoc local classifiers. In *Natural Language Processing and Information Systems*, Lecture Notes in Computer Science, 2023.

[32] Sadeq Darrab, Harshitha Allipilli, Sana Ghani, Harikrishnan Changaramkulath, Sricharan Koneru, David Broneske, and Gunter Saake. Anomaly detection algorithms: Comparative analysis and explainability perspectives. In Diana Benavides-Prado, Sarah M. Erfani, Philippe Fournier-Viger, Yee Ling Boo, and Yun Sing Koh, editors, *Data Science and Machine Learning - 21st Australasian Conference, AusDM 2023, Auckland, New Zealand, December 11-13, 2023, Proceedings*, volume 1943 of *Communications in Computer and Information Science*, pages 90–104. Springer, 2023.

[33] Ryota Nozawa, Shun Sato, and Takayasu Matsuo. A novel interpretation of nesterov's acceleration via variable step-size linear multistep methods. *CoRR*, abs/2404.10238, 2024.

[34] Alexander Vosseler. Unsupervised insurance fraud prediction based on anomaly detector ensembles. *Risks*, 2022.

[35] Mononito Goswami, Cristian Challu, Laurent Callot, Lenon Minorics, and Andrey Kan. Unsupervised model selection for time-series anomaly detection. *ArXiv*, 2022.

[36] Chunxu Cao and Qiang Zhang. A contrast based feature selection algorithm for high-dimensional data set in machine learning. *CoRR*, abs/2401.07482, 2024.

[37] Haotian Chang, Jing Feng, Chaofan Duan, Chao Yan, Min Yin, and Yi Li. A novel framework for anomaly detection via feature selection and dimensionality reduction. In *Fuzzy Systems and Data Mining V - Proceedings of FSDM 2019, Kitakyushu City, Japan, October 18-21, 2019*, volume 320 of *Frontiers in Artificial Intelligence and Applications*, pages 511–522, 2019.

[38] Wei Guo, Zhe Wang, Hai Yang, and Wenli Du. Multi-view dimensionality reduction learning with hierarchical sparse feature selection. *Appl. Intell.*, 53(10):12774–12791, 2023.

[39] Lucas Costa Brito, Gian Antonio Susto, Jorge Nei Brito, and Marcus Antonio Viana Duarte. Fault detection of bearing: An unsupervised machine learning approach exploiting feature extraction and dimensionality reduction. *Informatics*, page 85, 2021.

[40] Yulin Guo, S. Mahadevan, Shunsaku Matsumoto, Shunsuke Taba, and Daigo Watanabe. Investigation of surrogate modeling options with high-dimensional input and output. *AIAA Journal*, 2023.

[41] Yisroel Mirsky, Tomer Doitshman, Yuval Elovici, and Asaf Shabtai. Kitsune: An ensemble of autoencoders for online network intrusion detection. In *Annual Network and Distributed System Security Symposium (NDSS)*, 2018.

[42] Ruming Tang, Zheng Yang, Zeyan Li, Weibin Meng, Haixin Wang, Qi Li, Yongqian Sun, Dan Pei, Tao Wei, Yanfei Xu, and Yan Liu. Zerowall: Detecting zero-day web attacks through encoder-decoder recurrent neural networks. In *IEEE Conference on Computer Communications (INFOCOM)*, 2020.

[43] Chuanpu Fu, Qi Li, Meng Shen, and Ke Xu. Realtime robust malicious traffic detection via frequency domain analysis. In *ACM SIGSAC Conference on Computer and Communications Security (CCS)*, 2021.

[44] Leo Breiman, J. H. Friedman, R. A. Olshen, and C. J. Stone. *Classification and Regression Trees*. 1984.

[45] Christian Camilo Urcuqui López, Jose Osorio Quintero, Melisa García Pea, and Andres Navarro. Malicious and benign websites. 2017.

[46] Salvatore J Stolfo, Wei Fan, Wenke Lee, Andreas Prodromidis, and Philip K Chan. Cost-based modeling and evaluation for data mining with application to fraud and intrusion detection: Results from the jam project. 1999.

[47] Jingyu Xiao, Qingsong Zou, Qing Li, Dan Zhao, Kang Li, Wenxin Tang, Runjie Zhou, and Yong Jiang. User device interaction prediction via relational gated graph attention network and intent-aware encoder. In *Proceedings of the 2023 International Conference on Autonomous Agents and Multiagent Systems (AAMAS)*, pages 1634–1642, 2023.

[48] Lisa Liu, Gints Engelen, Timothy M. Lynar, Daryl Essam, and Wouter Joosen. Error prevalence in NIDS datasets: A case study on CIC-IDS-2017 and CSE-CIC-IDS-2018. In *IEEE Conference on Communications and Network Security (CNS)*, 2022.

[49] Tim M. Booij, Irina Chiscop, Erik Meeuwissen, Nour Moustafa, and Frank T. H. den Hartog. Ton_iot: The role of heterogeneity and the need for standardization of features and attack types in iot network intrusion data sets. *IEEE Internet of Things Journal*, 9(1):485–496, 2022.

[50] Xing Xu, Jie Li, Yang Yang, and Fumin Shen. Toward effective intrusion detection using log-cosh conditional variational autoencoder. *IEEE Internet of Things Journal*, pages 6187–6196, 2021.

[51] Adel Binbusayyis and Thavavel Vaiyapuri. Unsupervised deep learning approach for network intrusion detection combining convolutional autoencoder and one-class SVM. *Appl. Intell.*, 51(10):7094–7108, 2021.

[52] Yutao Dong, Qing Li, Kaidong Wu, Ruoyu Li, Dan Zhao, Gareth Tyson, Junkun Peng, Yong Jiang, Shutao Xia, and Mingwei Xu. Horuseye: Realtime iot malicious traffic detection framework with programmable switches. In *USENIX Security Symposium*, 2023.

[53] João Pereira and Margarida Silveira. Unsupervised anomaly detection in energy time series data using variational recurrent autoencoders with attention. In M. Arif Wani, Mehmed M. Kantardzic, Moamar Sayed Mouchaweh, João Gama, and Edwin Lughofer, editors, *17th IEEE International Conference on Machine Learning and Applications, ICMLA 2018, Orlando, FL, USA, December 17-20, 2018*, pages 1275–1282. IEEE, 2018.

[54] Bo Zong, Qi Song, Martin Renqiang Min, Wei Cheng, Cristian Lumezanu, Dae-ki Cho, and Haifeng Chen. Deep autoencoding gaussian mixture model for unsupervised anomaly detection. In *6th International Conference on Learning Representations, ICLR 2018, Vancouver, BC, Canada, April 30 - May 3, 2018, Conference Track Proceedings*. OpenReview.net, 2018.

[55] Alberto Barbado, Oscar Corcho, and Richard Benjamins. Rule extraction in unsupervised anomaly detection for model explainability: Application to oneclass svm. *Expert Systems with Applications*, 189:116100, 2022.

[56] Yiming Li, Jiawang Bai, Jiawei Li, Xue Yang, Yong Jiang, and Shu-Tao Xia. Rectified decision trees: Exploring the landscape of interpretable and effective machine learning. *CoRR*, abs/2008.09413, 2020.

[57] Giulia Vilone, Lucas Rizzo, and Luca Longo. A comparative analysis of rule-based, model-agnostic methods for explainable artificial intelligence. In *Irish Conference on Artificial Intelligence and Cognitive Science*, 2020.

[58] Nidula Elgiriyewithana. Credit card fraud detection dataset 2023, 2023.

[59] Euclides Carlos Pinto Neto, Sajjad Dadkhah, Raphael Ferreira, Alireza Zohourian, Rongxing Lu, and Ali A. Ghorbani. Ciciot2023: A real-time dataset and benchmark for large-scale attacks in iot environment. *Sensors*, 23(13):5941, 2023.

[60] Bushra A. AlAhmadi, Louise Axon, and Ivan Martinovic. 99% false positives: A qualitative study of SOC analysts' perspectives on security alarms. In *USENIX Security Symposium*, 2022.

# A Appendix

## A.1 Proof of Proposition

Let $\mathbf{X} \subset \mathbb{R}^d$ be a dataset sampled from a stationary distribution $\mathcal{D}$, and let $f : \mathbf{X} \to \{0, 1\}$ be an unsupervised anomaly detection model that assigns an anomaly score to each data point $\mathbf{x} \in \mathbf{X}$. There exists a rule-based surrogate model $h_{\mathcal{R}}(\mathbf{x})$ derived from $f$ such that the surrogate model maintains high fidelity to the original model. Specifically, the surrogate model achieves a high true negative rate (TNR) and true positive rate (TPR) compared to the original model.

**Proposition:** *The rule-based surrogate model $h_{\mathcal{R}}(\mathbf{x})$ derived from $f$ achieves high fidelity, specifically:*

$$\mathcal{L}(\hat{y}_f, \hat{y}_h) \leq \epsilon, \tag{9}$$

**Our Goal:** We aim to demonstrate that the rule-based surrogate model $h_{\mathcal{R}}(\mathbf{x})$ derived from the original anomaly detection model $f$ has high fidelity, meaning it closely approximates the behavior of $f$ in terms of TNR and TPR.

*where $\mathcal{L}$ is a loss function measuring the discrepancy between the predictions of $f$ and $h_{\mathcal{R}}$, and $\epsilon$ is a small positive constant.*

**Proof:**

1. **Rule Extraction:** The rules $\mathcal{R}$ are extracted from the original model $f$ by analyzing the decision boundaries defined by $f$. Each rule $R_i \in \mathcal{R}$ is of the form:

$$R_i = \bigwedge_{j=1}^{d} (x_j \odot_j \varphi_{ij}), \tag{10}$$

   where $\odot_j \in \{\leq, >\}$ is the relational operator, and $\varphi_{ij}$ is the threshold for the $j$-th feature in the $i$-th rule.

2. **Surrogate Model Construction:** The surrogate model $h_{\mathcal{R}}(\mathbf{x})$ is constructed using the rule set $\mathcal{R}$. A data point $\mathbf{x}$ is classified as normal if it satisfies at least one rule $R_i \in \mathcal{R}$:

$$h_{\mathcal{R}}(\mathbf{x}) = \neg \bigwedge_{R \in \mathcal{R}} (\mathbf{x} \in R). \tag{11}$$

3. **Fidelity Measurement:** Fidelity is measured by the loss function $\mathcal{L}$, which captures the discrepancy between the predictions of $f$ and $h_{\mathcal{R}}$. A common choice for $\mathcal{L}$ is the misclassification rate:

$$\mathcal{L}(\hat{y}_f, \hat{y}_h) = \frac{1}{|\mathbf{X}|} \sum_{\mathbf{x} \in \mathbf{X}} |\hat{y}_f - \hat{y}_h|. \tag{12}$$

4. **True Negative Rate (TNR) and True Positive Rate (TPR):** Define TNR and TPR for $f$ and $h_{\mathcal{R}}$:

$$\text{TNR}_f = \frac{\sum_{\mathbf{x} \in \mathbf{X}, y=0} I(f(\mathbf{x}) = 0)}{\sum_{\mathbf{x} \in \mathbf{X}, y=0} 1}, \quad \text{TPR}_f = \frac{\sum_{\mathbf{x} \in \mathbf{X}, y=1} I(f(\mathbf{x}) = 1)}{\sum_{\mathbf{x} \in \mathbf{X}, y=1} 1}, \tag{13}$$

$$\text{TNR}_h = \frac{\sum_{\mathbf{x} \in \mathbf{X}, y=0} I(h_{\mathcal{R}}(\mathbf{x}) = 0)}{\sum_{\mathbf{x} \in \mathbf{X}, y=0} 1}, \quad \text{TPR}_h = \frac{\sum_{\mathbf{x} \in \mathbf{X}, y=1} I(h_{\mathcal{R}}(\mathbf{x}) = 1)}{\sum_{\mathbf{x} \in \mathbf{X}, y=1} 1}. \tag{14}$$

5. **High Fidelity:** To achieve high fidelity, we need $\text{TNR}_h$ and $\text{TPR}_h$ to be close to $\text{TNR}_f$ and $\text{TPR}_f$, respectively. Formally, we want:

$$|\text{TNR}_f - \text{TNR}_h| \leq \delta \quad \text{and} \quad |\text{TPR}_f - \text{TPR}_h| \leq \delta, \tag{15}$$

   where $\delta$ is a small positive constant.

6. **Bounding the Loss:** Since $\hat{y}_h$ is derived from $\mathcal{R}$, which closely follows the decision boundaries of $f$, the discrepancy between $\hat{y}_f$ and $\hat{y}_h$ should be minimal. Thus, the loss function $\mathcal{L}$ can be bounded:

$$\mathcal{L}(\hat{y}_f, \hat{y}_h) \leq \frac{1}{|\mathbf{X}|} \left( \sum_{\mathbf{x} \in \mathbf{X}, y=0} |\text{TNR}_f - \text{TNR}_h| + \sum_{\mathbf{x} \in \mathbf{X}, y=1} |\text{TPR}_f - \text{TPR}_h| \right). \quad (16)$$

7. **Small Error Term:** Given the small values of $\delta$, the overall loss can be bounded by:

$$\mathcal{L}(\hat{y}_f, \hat{y}_h) \leq \epsilon, \quad (17)$$

$$\epsilon = \delta \cdot \left( \frac{\sum_{\mathbf{x} \in \mathbf{X}, y=0} 1}{|\mathbf{X}|} + \frac{\sum_{\mathbf{x} \in \mathbf{X}, y=1} 1}{|\mathbf{X}|} \right) \quad (18)$$

8. **Finally:** The rule-based surrogate model $h_{\mathcal{R}}(\mathbf{x})$ derived from the original anomaly detection model $f$ achieves high fidelity, with the discrepancy between the two models being bounded by a small constant $\epsilon$.

## A.2 Algorithmic implementation

In Algorithm 1, we combines SCD-Tree and GPR to establish decision boundaries for anomaly detection in high-dimensional data. Initially, the anomaly detector $f$ gets a sample anomaly score $s$ and a threshold $\varphi$ to classify instances in the dataset $\mathbf{X}$ as normal and abnormal. The SCD-Tree then recursively segments the dataset into distributions based on these scores, using information gain to determine optimal features and thresholds for splitting.

Within each distribution $\mathbf{X}_k$, a Gaussian Process with an RBF kernel models the data in each dimension. After preprocessing and normalizing the data, the GP predicts on an extended range to determine where normality changes, updating decision boundaries based on the longest interval meeting the normality criteria. These boundaries are then formulated into rules $\mathcal{R}_k$, each a conjunction of dimensional conditions, enhancing interpretability and the utility of anomaly detection models.

---

**Algorithm 1** Implementation of SCD-Tree and Gaussian Process for Boundary Delineation

**Input:** $\mathbf{X}$ as the full dataset, anomaly detector $f$
**Output:** Boundary inference rule $\mathcal{R}_k$ on this leaf node, which encapsulates normality

1 (score $s$, threshold $\varphi$) $\leftarrow f(X)$

2 /* Build Segmentation Clustering Decision Tree*/
3 **foreach** *node $N$* in SCD-Tree from root to leaves **do**
4     $A \leftarrow$ Select feature and threshold for split based on anomaly scores; $\triangleright$ Select best attribute based on information gain
        Split $N$ into child nodes $N_{\text{left}}$ and $N_{\text{right}}$ using $A$;        $\triangleright$ Divide data into two subsets
        Recursive splitting until all leaf nodes are obtained
5 **end foreach**

6 /* Train Gaussian Process Regression model*/
7 **for** *each cluster $\mathbf{X}_k$ of leaf node $N_k$* **do**
8     Initialize boundaries for $\mathbf{X}_k$;      $\triangleright$ Set initial decision boundaries for normal data within node
9     **for** *each dimension $i$ in $\mathbf{X}_k$* **do**
10        $X\_input \leftarrow$ preprocess($X\_inlier_k$);     $\triangleright$ Prepare data for GP by normalizing or scaling
11        $gp \leftarrow$ **GPRegression**($X\_input$, ($y_k \leftarrow (s,\varphi)$), **RBF**kernel);
12        $samples \leftarrow$ extend_scope($X\_input$);     $\triangleright$ Generate expanded sample range for prediction
13        $y_{\text{pred}}, \sigma \leftarrow$ gp.**predict**($samples$);     $\triangleright$ Predict with GP to estimate boundary regions
14        $interval \leftarrow$ find_longest_interval($y_{\text{pred}}, \sigma$);    $\triangleright$ Identify longest interval meeting criteria for normality
15        $r_i \leftarrow$ update_boundaries($interval$) **if** interval exists
16    **end for**
17    $\mathcal{R}_k \leftarrow (r_1 \wedge r_2 \wedge \ldots \wedge r_d)$;     $\triangleright$ Combine all dimensional criteria to form distribution rule
18 **end for**
19 **return** $\mathcal{R}_k$;        $\triangleright$ Output boundary rules for all clusters

---

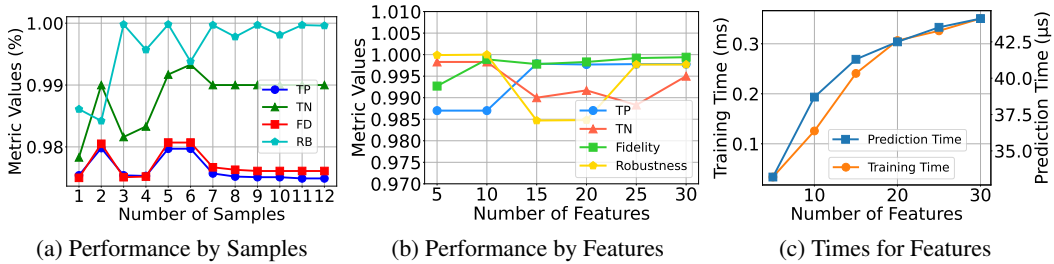

(a) Performance by Samples     (b) Performance by Features     (c) Times for Features

Figure 2: More Experiment Details

## A.3 Implementation of experiment

**Software and Computational Framework.** Our implementation utilizes PyTorch (version 2.1.0) to facilitate the development and training of deep learning models, specifically Autoencoders (AE) and Variational Autoencoders (VAE), which are crucial for our anomaly detection tasks. Complementing this, scikit-learn (version 1.1.3) is employed for essential preprocessing, feature engineering, and the evaluation of models. The entire system is orchestrated using Python (version 3.8.18), chosen for its extensive libraries that streamline data manipulation and experimental workflows.

**Hardware Specifications.** The computational infrastructure is centered around a high-capacity server featuring an Intel(R) Xeon(R) Gold 5318Y CPU @ 2.10GHz with 527GB of RAM, supporting the intensive computational demands of our experiments. Additionally, an NVIDIA GeForce RTX 3090 Super with 24GB VRAM is utilized specifically for the computationally intensive tasks of training our deep learning models. Notably, the extraction of rules, a CPU-intensive process, highlights the efficiency and adaptability of our implementation in resource allocation.

**Experiment Details.** The deployment of our models and the execution of experiments adhere to a structured approach, ensuring reproducibility and systematic analysis. Details on model hyperparameters, dataset splits, and the random seeds used to ensure reproducibility will be available through an anonymous repository, providing full access to the scripts and setup used in our research.

## A.4 More Experiments

### A.4.1 Hyperparameter

We detail the hyperparameter tuning process for the SCD-Tree and GBD algorithms, focusing on the influence of feature count and sample size on performance metrics.

**Impact of Feature Count on Performance Metrics:** To understand the influence of the number of features on the performance of our proposed method, we conducted experiments varying the feature count from 5 to 30. The results in Figure 2b show that the true positive and true negative rates are consistently high for different numbers of features, and can remain highly accurate even when interpreting very few features This indicates that the model maintains a high detection accuracy regardless of the number of features. Fidelity displayed an increasing trend with the number of features, suggesting that the surrogate model's ability to approximate the black-box model improves with more features. Robustness was generally high, with minor variations, reflecting the model's stability under different feature configurations.

**Impact of Sample Size on Performance Metrics:** We also evaluated the performance of our method across varying numbers of expanded samples[3] in the GP algorithm, ranging from 1 to 12. The results indicated that TP and TN rates remained relatively stable across different expanded sample sizes, indicates that while even a small number of expanded samples can achieve good accuracy. In addition, Fidelity remained high across all expanded sample sizes, showing the surrogate model's consistent approximation of the black-box model in handling varying levels of data expansion. Notably, Robustness saw a significant increase as the number of expanded samples increased.

### A.4.2 Complexity testing

We delve into the computational cost and complexity associated with training our model on the CIC-IDS2017 dataset using a VAE as black-box model, investigating the impact of varying the number of features on the training time. The training time increases almost linearly with the number of features as Figure 2c shows. This trend aligns with theoretical expectations, as the addition of each feature introduces more parameters that the VAE must learn. Specifically, the training time grows from 0.03353 ms for 5 features to 0.3504 ms for 30 features. The increase in training time with more features is due to the additional complexity in splitting and boundary delineation within the SCD-Tree and GBD algorithms.

As the number of features increases, the model reaches a point where additional features contribute less to the overall complexity. This phenomenon is due to the fact that not all features add unique information. Some features might be redundant or highly correlated with others, providing little additional value for the model to learn.The SCD-Tree's ability to segment the data may reach an optimal level of complexity. After a certain number of features, further splits may not be necessary or as impactful, resulting in fewer additional splits and less computational overhead. The tree may also prune less informative features, focusing on the most relevant ones, thereby maintaining efficiency.

Despite the observed increase in training time, our model maintains highly efficient prediction times. This efficiency is achieved by leveraging a set of pre-defined rules derived from the SCD-Tree's decision paths and the GBD's boundary delineations. These rules are computationally inexpensive to evaluate, ensuring that our model remains suitable for real-time applications. The prediction process is streamlined and does not require the iterative optimization steps needed during training, thus significantly reducing computational overhead.

## B Model Performance

In this section, we present a detailed analysis of the performance of our proposed method for rule-based interpretation in unsupervised anomaly detection models. The evaluation is based on a series of experiments conducted on the CIC-IDS2017 dataset and three additional datasets (KddCup, TON-IoT, and Web) using four different anomaly detection models: Autoencoder (AE), Variational Autoencoder (VAE), One-Class SVM (OCSVM), and Isolation Forest (iForest).

### B.1 Performance on CIC-IDS2017 Dataset

We first evaluated the performance of our method on the CIC-IDS2017 dataset. Table 5 summarizes the results for six different methods: UAD, EGDT, Trustee, LIME, KD, and our proposed method. The metrics used for evaluation include Classification Rate (CR), Precision (PR), Recall (RC), and F1 score (F1). Our method demonstrates superior performance across all metrics compared to the baseline methods. Specifically, with the VAE as the black-model, our method achieves a CR of 0.9975, PR of 0.9992, RC of 0.9979, and F1 score of 0.9994, almost all metrics are at the optimal value for all baselines. Similar trends are observed for the VAE, OCSVM, and iForest models, indicating the robustness and effectiveness of our approach in extracting interpretable rules that closely approximate the original model's behavior.

Table 5: Performance of rule extraction on CIC-IDS2017 dataset.

| Method | AE | | | | VAE | | | | OCSVM | | | | iForest | | | |
|---|---|---|---|---|---|---|---|---|---|---|---|---|---|---|---|---|
| | CR | PR | RC | F1 | CR | PR | RC | F1 | CR | PR | RC | F1 | CR | PR | RC | F1 |
| UAD | 0.0135 | 0.9997 | 0.0118 | 0.0231 | 0.6096 | 0.0866 | 0.1124 | 0.1903 | 0.9972 | 0.9997 | 0.996 | 0.9977 | 0.9988 | 0.0677 | 0.9299 | 0.9623 |
| EGDT | 0.5731 | 0.4726 | 0.4734 | 0.8932 | 0.5617 | 0.8724 | 0.6859 | 0.5455 | 0.9898 | 0.925 | 0.8544 | 0.9185 | 0.9949 | 0.9393 | 0.9938 | 0.9938 |
| Trustee | 0.8652 | 0.4248 | 0.6479 | 0.4248 | 0.8123 | 0.8319 | 0.7663 | 0.8278 | 0.8551 | 0.5509 | 0.666 | 0.7502 | 0.9549 | 0.4847 | 0.8673 | 0.8947 |
| LIME | 0.9065 | 1 | 0.9162 | 0.9493 | 0.9063 | 0.9996 | 0.916 | 0.9492 | 0.9883 | 0.7413 | 0.9263 | 0.9574 | 0.8644 | 0.8305 | 0.8296 | 0.8759 |
| KD | 0.5147 | 0.5143 | 0.6863 | 0.5832 | 0.623 | 0.9823 | 0.4799 | 0.6448 | 0.9154 | 0.3565 | 0.3744 | 0.5314 | 0.7853 | 0.0666 | 0.8463 | 0.8466 |
| Ours | 0.9438 | 0.999 | 0.936 | 0.9665 | **0.9975** | 0.9992 | **0.9979** | **0.9986** | 0.9707 | 0.9995 | 0.9666 | 0.9828 | 0.9717 | **0.9994** | 0.9678 | 0.9834 |

## B.2 Performance on Different Datasets

To further assess the generalizability of our method, we conducted experiments on three additional datasets: KddCup, TON-IoT, and Web, as summarized in Table 6. Our method consistently outperformed baseline approaches across all datasets and metrics. On the KddCup dataset, for instance, we achieved F1 scores of 0.9781, 0.9848, 0.9342, and 0.9901 for the AE, VAE, OCSVM, and iForest models, respectively. This highlights the method's adaptability and accuracy across different models and complex anomaly detection scenarios.

In addition, Table 7 compares the performance of our rule extraction method using VAE as a black-box model across different domain datasets, including Credit Card Fraud Detection, CIC-IoT, and Breast Cancer Wisconsin. Across these datasets, our method consistently demonstrated superior True Positive Rates (TPR) and True Negative Rates (TNR), indicating its effectiveness in both detecting anomalies and correctly identifying normal instances. For example, on the Credit Card Fraud dataset, we achieved a TPR of 0.9127 and a TNR of 0.9854, demonstrating the method's reliability in handling imbalanced data. The high fidelity and robustness across diverse datasets confirm the effectiveness of our approach, even in high-stakes domains like healthcare and IoT security. Overall, the results from both tables underscore the versatility, accuracy, and robustness of our method, validating its applicability across different anomaly detection scenarios and datasets.

Table 6: Performance of rule extraction on different datasets.

| Datasets | AE | | | | VAE | | | | OCSVM | | | | iForest | | | |
|---|---|---|---|---|---|---|---|---|---|---|---|---|---|---|---|---|
| | CR | PR | RC | F1 | CR | PR | RC | F1 | CR | PR | RC | F1 | CR | PR | RC | F1 |
| CIC-IDS | 0.9438 | 0.999 | 0.936 | 0.9665 | 0.9975 | 0.9992 | 0.9979 | 0.9986 | 0.9707 | 0.9995 | 0.9666 | 0.9828 | 0.9717 | 0.9994 | 0.9678 | 0.9834 |
| KddCup | 0.9635 | 0.9576 | 0.9994 | 0.9781 | 0.976 | 0.9968 | 0.9731 | 0.9848 | 0.894 | 0.9284 | 0.94 | 0.9342 | 0.9905 | 0.9835 | 0.9969 | 0.9901 |
| TON-IoT | 0.9856 | 0.9759 | 0.9679 | 0.9719 | 0.9917 | 0.9689 | 1 | 0.9842 | 0.9955 | 0.9829 | 1 | 0.9914 | 0.9991 | 0.9963 | 1 | 0.9982 |
| Web | 0.997 | 1 | 0.88 | 0.9362 | 0.996 | 1 | 0.8462 | 0.9167 | 0.998 | 1 | 0.9184 | 0.9574 | 0.9955 | 1 | 0.8235 | 0.9032 |

Table 7: Performance of Rule Extraction Across Selected Security Datasets

| Method | Credit Card Fraud Detection [58] | | | | CIC-IoT [59] | | | | Breast Cancer Wisconsin | | | |
|---|---|---|---|---|---|---|---|---|---|---|---|---|
| | TPR | TNR | FD | RB | TPR | TNR | FD | RB | TPR | TNR | FD | RB |
| UAD | 0.6256 | 0.8331 | 0.7861 | 0.5021 | 0.8913 | 0.9988 | 0.3644 | 0.4834 | 0.2632 | 0.0132 | 0.114 | 0.9912 |
| LIME | 0.8951 | 0.9742 | 0.9156 | 0.9985 | 0.9063 | 0.9786 | 0.9410 | 0.9996 | 0.6579 | 0.9474 | 0.8333 | 1 |
| Trustee | 0.5325 | 0.9923 | 0.6985 | 0.9910 | 0.4165 | 0.9972 | 0.8126 | 0.9929 | 0.1316 | 0.9868 | 0.6842 | 0.9825 |
| Ours | 0.9127 | 0.9854 | 0.9335 | 0.8964 | 0.9150 | 0.9899 | 0.9453 | 0.8979 | 0.8684 | 0.75 | 0.8421 | 1 |

## B.3 Model validity Experiment

Table 8: The number and average length of rules generated by the model

| Dataset | Number of Rules | | | Average Rule Length | | |
|---|---|---|---|---|---|---|
| | AE | VAE | IFOREST | AE | VAE | IFOREST |
| CIC-IDS | 22 | 15 | 17 | 4.83 | 3.03 | 4.97 |
| ton-iot | 21 | 23 | 13 | 5 | 5 | 5.0 |
| kddcup | 17 | 19 | 21 | 5 | 4.7 | 5 |

In our study, we have emphasized the significance of interpretability in anomaly detection models, especially in high-stakes domains such as cybersecurity, where comprehensibility of model decisions is crucial for timely and accurate responses. We use the model to test the rules that can be generated on different datasets. For the CIC-IDS dataset, the Autoencoder generated 22 rules with an average rule length of 4.83, indicating a relatively detailed set of explanations that captures the complexity of network intrusion patterns. The VAE model produced a more concise rule set, with 15 rules and an average length of 3.03, suggesting that the VAE may encapsulate the data distribution with fewer, simpler rules while maintaining interpretability.

### B.4 Analysis of Results

The experimental results clearly indicate the superiority of our proposed method in terms of interpretability and fidelity to the original anomaly detection models. The high values of CR, PR, RC, and F1 scores across different datasets and models confirm that our method can effectively extract rules that closely mimic the behavior of complex black-box models. This not only enhances the transparency of the anomaly detection process but also facilitates better understanding and trust in automated systems, particularly in high-stakes environments where decision accuracy is crucial.

In summary, our method provides a robust and interpretable framework for unsupervised anomaly detection, offering significant improvements over existing baseline methods. The ability to transform complex model operations into clear, rule-based explanations makes it a valuable tool for practitioners and researchers in the field.

### B.4.1 Metrics

In our evaluation, we utilize a set of metrics which include: (1) Fidelity, which measures the congruence between the predictions of the original model and its explanations to assess the reliability of those explanations; (2) Robustness, evaluating the consistency of explanations in the face of slight variations in input data; and (3) both True Positive Rate and True Negative Rate, which are critical for determining the practical effectiveness in security contexts where minimizing false alarms is pivotal to averting "alert fatigue" [60]. These metrics form a comprehensive framework for assessing the explanatory capacity and dependability of various approaches, ensuring that our results are solid and directly applicable to real-world anomaly detection scenarios. Their meaning and calculations are:

- **TNR and TPR** represent the probability of being correctly identified as positive (abnormal) and being correctly identified as negative (normal), respectively. The formulas are $TP = \frac{tp}{tp+fn}$, $TN = \frac{tn}{tn+fp}$. These metrics form the basis for more complex evaluation metrics. In the context of our model, these values are derived from the confusion matrix, which summarizes the performance of the classification.

- **Fidelity (FD)** measures the extent to which the rule-based explanations generated by our method agree with the predictions made by the underlying black-box model. High fidelity indicates that the surrogate model (SCD-Tree + GBD) closely approximates the decision-making process of the black-box model, defined as $FD = \frac{\text{Number of consistent predictions}}{\text{Total number of predictions}}$. Let $h_{\mathcal{R}}(x)$ be the surrogate model based on the extracted rules $\mathcal{R}$. Define fidelity as the agreement between the surrogate model and the original model $f$: $\mathcal{FD} = \frac{1}{n}\sum_{i=1}^{n}[h_{\mathcal{R}}(x_i) = f(x_i)]$ The rules are derived directly from the KITree's splits and the GP's boundary delineation, ensuring that $h_{\mathcal{R}}(x)$ closely follows $f$.

- **Robustness (RB)** evaluates the stability of the model's explanations under slight perturbations in the input data. It assesses whether the explanations remain consistent when the input data is subject to small changes, which is crucial for ensuring the reliability of the model in dynamic environments, assessed by $RB = \frac{\text{Number of consistent predictions under perturbation}}{\text{Total number of predictions}}$. Consider small perturbations $\delta$ in the input $x$: $x' = x + \delta$ Robustness is defined as the invariance of predictions under these perturbations: $RB = \frac{1}{n}\sum_{i=1}^{n}[h_{\mathcal{R}}(x_i) = h_{\mathcal{R}}(x_i + \delta)$. The decision boundaries are derived from GPs, which provide smooth and continuous estimates, inherently leading to robust predictions.

### B.5 Limitations

While our method introduces significant advancements in the interpretability of unsupervised anomaly detection models, several limitations warrant further exploration. Primarily, the efficacy of our SCD-Tree and GBD algorithms hinges predominantly on structured, tabular data, where each feature $x_i$ within the vector $\mathbf{x} \in \mathbb{R}^d$ conveys explicit semantic information. This specificity restricts direct applicability to complex, unstructured data forms such as raw images or sequential text, where high-level feature extraction typically necessitates convolutional neural networks (CNNs) or transformers to preprocess data into an analyzable format for our model. For instance, if $X$ represents an image, direct application of our rule extraction process becomes nonviable without a preceding transformation $T : \mathcal{X}_{\text{raw}} \rightarrow \mathcal{X}_{\text{structured}}$, where $\mathcal{X}_{\text{raw}}$ and $\mathcal{X}_{\text{structured}}$ denote the spaces of raw and structured data, respectively.

Furthermore, the extracted rules are inherently axis-aligned, constraining their ability to encapsulate complex, non-linear decision boundaries prevalent in various applications. If $\mathcal{B}(\mathbf{x})$ represents the original model's boundary function in a high-dimensional space, the limitation in capturing its curvature through axis-aligned splits can be expressed as:

$$\min_{\mathcal{R}} \left\{ \sum_{\mathbf{x} \in \mathcal{D}} |\mathcal{B}(\mathbf{x}) - \mathcal{R}(\mathbf{x})| \right\} \tag{19}$$

where $\mathcal{R}$ signifies the set of rules derived from our method, and $\mathcal{D}$ denotes the data distribution. While the introduction of compositional distribution segmentation within our SCD-Tree mitigates this issue partially, perfect congruence with irregular boundaries remains elusive, illustrating a common challenge across global explanation methodologies.

Lastly, the validation of our method's interpretability and trustworthiness, especially in domains heavily reliant on expert judgment, lacks empirical rigor. Future studies should emphasize structured user tests involving domain-specific practitioners to assess whether the interpretations generated align with expert reasoning and decision-making processes. Such evaluations could significantly augment our understanding and optimization of the interaction between human experts and automated systems, ideally leading to more intuitive and effective anomaly detection tools.

By addressing these areas, future research can expand the method's applicability, enhance its adaptability to complex data forms, and refine its utility in practical, expert-driven environments, ensuring broader adoption and deeper trust in automated anomaly detection.

